# DOES THE NEURON "LEARN" LIKE THE SYNAPSE?

RAOUL TAWEL

Jet Propulsion Laboratory
California Institute of Technology
Pasadena, CA 91109

**Abstract.** An improved learning paradigm that offers a significant reduction in computation time during the supervised learning phase is described. It is based on extending the role that the neuron plays in artificial neural systems. Prior work has regarded the neuron as a strictly passive, non-linear processing element, and the synapse on the other hand as the primary source of information processing and knowledge retention. In this work, the role of the neuron is extended insofar as allowing its parameters to adaptively participate in the learning phase. The temperature of the sigmoid function is an example of such a parameter. During learning, both the synaptic interconnection weights $w_{ij}^m$ and the neuronal temperatures $T_i^m$ are optimized so as to capture the knowledge contained within the training set. The method allows each neuron to possess and update its own characteristic local temperature. This algorithm has been applied to logic type of problems such as the XOR or parity problem, resulting in a significant decrease in the required number of training cycles.

## INTRODUCTION

One of the current issues in the theory of supervised learning concerns the scaling properties of neural networks. While low-order neural computations are easily handled on sequential or parallel processors, high-order problems prove to be intractable. The computational burden involved in implementing supervised learning algorithms, such as back-propagation, on networks with large connectivity and/or large training sets is immense and impractical at present. Therefore the treatment of 'real' applications in such areas as image recognition or pattern classification require the development of computationally efficient learning rules. This paper reports such an algorithm.

Current neuromorphic models regard the neuron as a strictly passive non-linear element, and the synapse on the other hand as the primary source of knowledge retention. In these models, information processing is performed by propagating the synaptically weighed neuronal contributions in either a feed forward, feed backward, or fully recurrent fashion [1]-[3]. Artificial neural networks commonly take the point of view that the neuron can be modeled by a simple non-linear 'wire' type of device.

However, evidence exists that information processing in biological neural networks does occur at the neuronal level [4]. Although neuromorphic nets based on simple neurons are useful as a first approximation, a considerable richness is to be gained by extending 'learning' to the neuron. In this work, such an extension is made. The neuron is then seen to provide an additional or secondary source of information processing and knowledge retention. This is achieved by treating both the neuronal and synaptic variables as optimization parameters. The temperature of the sigmoid function is an example of such a neuronal parameter. In much

the same way that the synaptic interconnection weights require optimization to reflect the knowledge contained within the training set, so should the temperature terms be optimized. It should be emphasized that the method does not optimize a global neuronal temperature for the whole network, but rather allows each neuron to posses and update its own characteristic local value.

## ADAPTIVE NEURON MODEL

Although the principle of neuronal optimization is an entirely general concept, and therefore applicable to any learning scheme, the popular feed forward back propagation (BP) learning rule has been selected for its implementation and performance evaluation. In this section we develop the mathematical formalism necessary to implement the adaptive neuron model (ANM).

Back propagation is an example of supervised learning where, for each presentation consisting of an input vector $\vec{o}^{1p}$ and its associated target vector $\vec{t}^p$, the algorithm attemps to adjust the synaptic weights so as to minimize the sum-squared error $E$ over all patterns $p$. In its simplest form, back propagation treats the interconnection weights as the only variable and consequently executes gradient descent in weight space. The error term is given by

$$E = \sum_p E_p = \frac{1}{2} \sum_p \sum_i [t_i^p - o_i^{np}]^2$$

The quantity $t_i^p$ is the $i^{th}$ component of the $p^{th}$ desired output vector pattern and $o_i^{np}$ is the activation of the corresponding neuron in the final layer $n$. For notational ease the summation over $p$ is dropped and a single pattern is considered. On completion of learning, the synaptic weights capture the transformation linking the input to output variables. In applications other than toy problems, a major drawback of this algorithm is the excessive convergence time.

In this paper it is shown that a significant decrease in convergence time can be realized by allowing the neurons to adaptively participate in the learning process. This means that each neuron is to be characterized by a set of parameters, such as temperature, whose values are optimized according to a rule, and not in a heuristic fashion as in simulated annealing. Upon training completion, learning is thus captured in both the synaptic and neuronal parameters.

The activation of a unit - say the $i^{th}$ neuron on the $m^{th}$ layer - is given by $o_i^m$. This response is computed by a non-linear operation on the weighed responses of neurons from the previous layer, as seen in Figure 1. A common function to use is the logistic funtion,

$$o_i^m = \frac{1}{1 + e^{-\beta s_i^m}}$$

and $T \equiv 1/\beta$ is the temperature of the network. The net weighed input to the neuron is found by summing products of the synaptic weights and corresponding neuronal ouputs from units on the previous layer,

$$s_i^m = \sum_j w_{ij}^{m-1} o_j^{m-1}$$

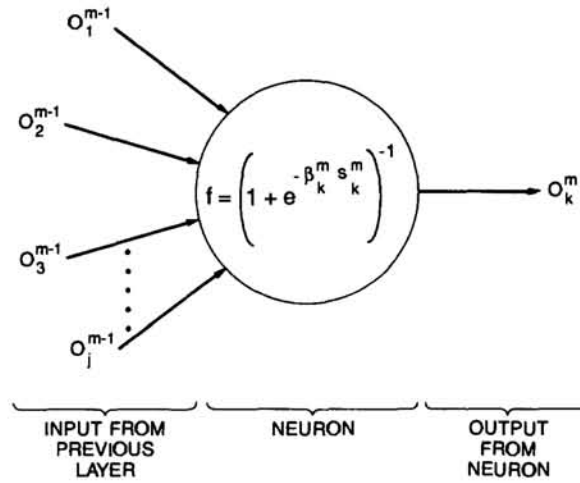

**Figure 1.** Each neuron in a network is chara-
terized by a local, temperature dependent, sig-
moidal activation function.

where $o_j^{m-1}$ represents fan-in units and the $w_{ij}^{m-1}$ represent the pairwise connection
strength between neuron $i$ in layer $m$ and neuron $j$ in layer $m-1$.

We have investigated several mathematical methods for the determination of the
optimal neuronal temperatures. In this paper, the rule that was selected to optimize
these parameters is based on executing gradient descent in the sum squared error
$E$ in temperature space. The method requires that the incremental change in the
temperature term be proportional to the negative of the derivative of the error term
with respect to the temperature. Focussing on the $l^{th}$ neuron on the ouput layer
$n$, we have

$$\Delta T_l^n = -\tilde{\eta} \frac{\partial E}{\partial T_l^n}$$

In this expression, $\tilde{\eta}$ is the temperature learning rate. This equation can be ex-
pressed as the product of two terms by the chain rule

$$\frac{\partial E}{\partial T_l^n} = \frac{\partial E}{\partial o_l^n} \frac{\partial o_l^n}{\partial T_l^n}$$

Substituting expressions and leaving the explicit functional form of the activation
function unspecified, i.e. $o_l^n = f(T_l^n, ...)$ we obtain

$$\frac{\partial E}{\partial T_l^n} = -[t_l - o_l^n] \frac{\partial f}{\partial T_l^n}$$

In a similar fashion, the temperature update equation for the previous layer is given
by,

$$\Delta T_k^{n-1} = -\tilde{\eta} \frac{\partial E}{\partial T_k^{n-1}}$$

Using the chain rule, this can be expressed as

$$\frac{\partial E}{\partial T_k^{n-1}} = \sum_l \frac{\partial E}{\partial o_l^n} \frac{\partial o_l^n}{\partial s_l^n} \frac{\partial s_l^n}{\partial o_k^{n-1}} \frac{\partial o_k^{n-1}}{\partial T_k^{n-1}}$$

Substituting expressions and simplifying reduces the above to

$$\frac{\partial E}{\partial T_k^{n-1}} = \left[ \sum_l -[t_l - o_l^n] \frac{\partial f}{\partial s_l^n} w_{lk}^{n-1} \right] \frac{\partial f}{\partial T_k^{n-1}}$$

By repeating the above derivation for the previous layer, i.e. determining the partial derivative of $E$ with respect to $T_j^{n-2}$ etc., a simple recursive relationship emerges for the temperature terms. Specifically, the updating scheme for the $k^{th}$ neuronal temperature on the $m^{th}$ layer is given by

$$\Delta T_k^m = -\tilde{\eta} \frac{\partial E}{\partial T_k^m}$$

where

$$\frac{\partial E}{\partial T_k^m} = -\delta_k^m \frac{\partial f}{\partial T_k^m}$$

In the above expression, the error signal $\delta_k^m$ takes on the value,

$$\delta_k^m = [t_k - o_k^m]$$

if neuron $m$ lies on an output layer, or

$$\delta_k^m = \sum_l \delta_l^{m+1} \frac{\partial f}{\partial s_l^{m+1}} w_{lk}^m$$

if the neuron lies on a hidden layer.

## SIMULATION RESULTS OF TEMPERATURE OPTIMIZATION

The new algorithm was applied to logic problems. The network was trained on a standard benchmark - the exclusive-or logic problem. This is a classic problem requiring hidden units and since many problems involve an XOR as a subproblem. As in plain BP, the application of the proposed learning rule involves two passes. In the first, an input pattern is presented and propagated forward through the network to compute the output values $o_j^n$ . This output is compared to its target value, resulting in an error signal for each output unit. The second pass involves a backward pass through the network during which the error signal is passed along the network and the appropriate weight and temperature changes made. Note that since the synapses and neurons have their own characteristic learning rate, i.e $\eta$ and $\tilde{\eta}$ respectively, an additional degree of freedom is introduced in the simulation. This is equivalent to allowing for relative updating time scales for the weights and

temperatures, i.e. $\tau_w$ and $\tau_T$ respectively. We have now generated a gradient descent method for finding weights and temperatures in a feed forward network.

In deriving the learning rule for temperature optimization in the above section, the derivative of the activation function of a neuron played a key role. We have used a sigmoidal type of function in our simulations whose explicit form is given by,

$$f\left(s_k^m, T_k^m\right) = \frac{1}{1 + e^{-\beta_k^m s_k^m}}$$

and in Figure 2 it is shown to be extremely sensitive to small changes in temperature.

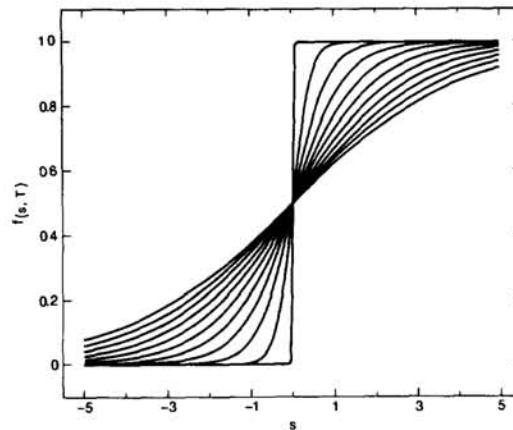

**Figure 2.** Activation function shown plotted for several different temperatures.

The sigmoid is shown plotted against the net input to a neuron for temperatures ranging from 0.2 to 2.0, in increments of 0.2. However, the steepest curve was for a temperature of 0.01. The derivative of the activation function taken with respect to the temperature is given by

$$\frac{\partial f}{\partial T_k^m} = -\frac{s_k^m}{T_k^{m\,2}} \frac{e^{-\beta_k^m s_k^m}}{\left(1 + e^{-\beta_k^m s_k^m}\right)^2}$$

As shown in Figure 3, the XOR architecture selected has two input units, two hidden units, and a single output unit. Each neuron is characterized by a temperature, and neurons are connected by weights. Prior to training the network, both the weights and temperatures were randomized. The initial and final optimization parameters for a sample training exercise are shown in Figure 3(a) & (b). Specifically, Figure 3(a) shows the values of the randomized weights and temperatures prior to training, and Figure 3(b) shows their values after training the network for 1000 iterations. This is a case where the network has reached a global minimum. In both figures, the numbers associated with the dashed arrows represent the thresholds of the neurons, and the numbers written next to the solid arrows represent the

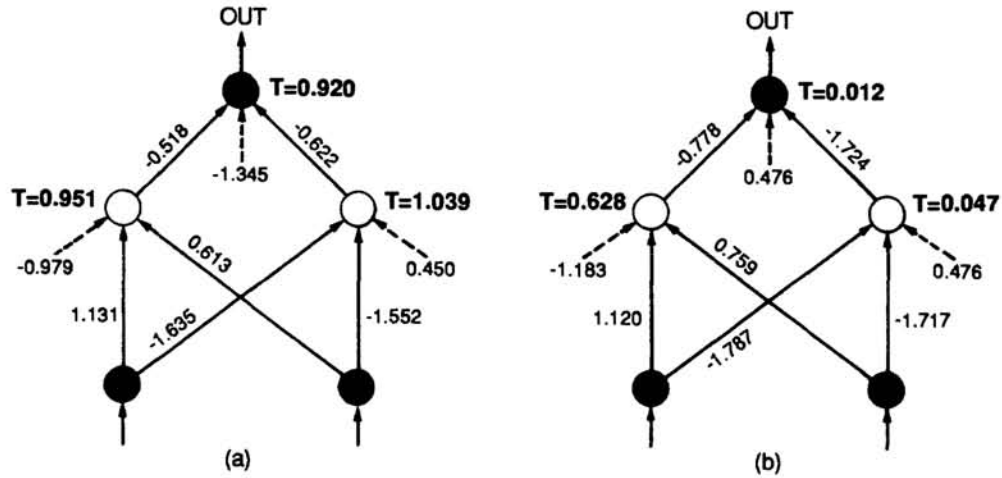

**Figure 3.** Architecture of NN for XOR problem showing neuronal temperatures and synaptic weights before (a) and after training (b).

excitatory/inhibitory strengths of the pairwise connections. To fully evaluate the convergence speed of the proposed algorithm, a benchmark comparison between it and plain BP was made. In both cases the training was started with identical initial random synaptic weights lying within the range $[-2.0, +2.0]$ and the same synaptic weight learning rate $\eta = 0.1$. The temperatures of the neurons in the ANM model were randomly selected to lie within the narrow range of $[0.9, 1.1]$ and the temperature learning rate $\tilde{\eta}$ set at 0.1. Figures 4(a) & (b) summarize the training statistics of this comparison.

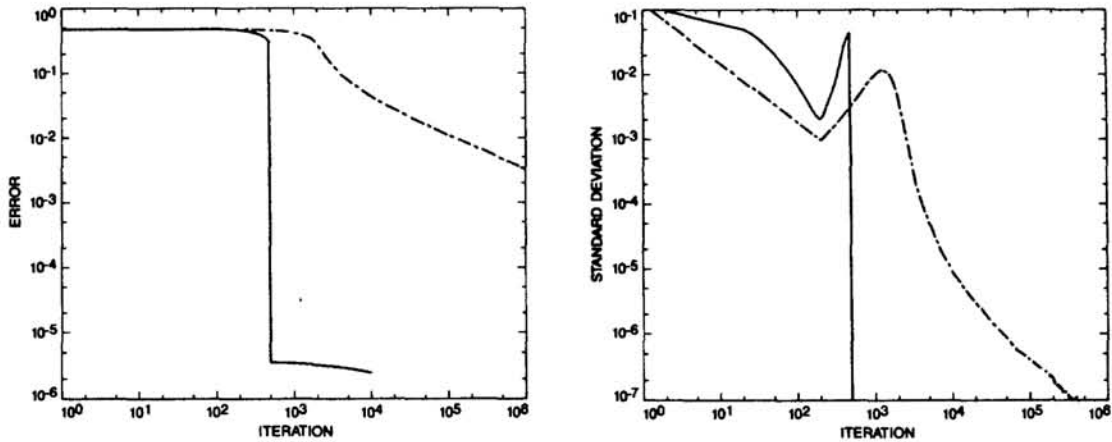

**Figure 4.** Comparison of training statistics between the adaptive neuron model and plain back propagation.

In both figures, the solid lines represent the ANM and the dashed lines represent the plain BP model. In Figure 4(a), the error is plotted against the training iteration number. In Figure 4(b), the standard deviation of the error over the training set is shown plotted against the training iteration. In the first few hundred training iterations in Figure 4(a), the performance of BP and the ANM is similar and appears as a broad shoulder in the curve. Recall that both the weights and temperatures are randomized prior to training, and are therefore far from their final values. As a consequence of the low values of the learning rates used, the error is large, and will only begin to get smaller when the weights and temperatures begin to fall in the right domain of values. In the ANM, the shoulder terminus is marked by a phase-transition like discontinuity in both error and standard deviation. For the particular example shown, this occured at the $637^{th}$ iteration. A several order of magnitude drop in the error and standard deviation is observed within the next 10 iterations. This sharp drop off is followed by a much more gradual decrease in both the error and standard deviation. A more detailed analysis of these results will be published in a longer paper.

In learning the XOR problem using standard BP, it has been observed that the network frequently gets trapped in local minima. In Figure 5(a) & (b) we observe such a case as shown by the dotted line. In numerous simulations on this problem, we have determined that the ANM is much less likely to become trapped in local minima.

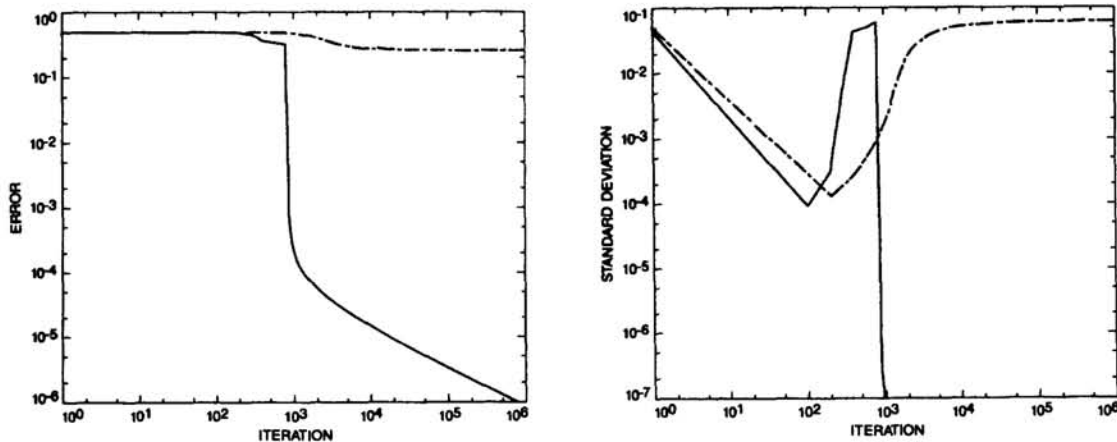

**Figure 5.** Training case where the adaptive neuron model escapes a local minima and plain back propagation does not.

## CONCLUSIONS

In this paper we have attempted to upgrade and enrich the model of the neuron from a simple static non-linear wire-type construct, to a dynamically reconfigurable one. From a purely computational point of view, there are definite advantages in such an extension. Recall that if $N$ is the number of neurons in a network then the number of synaptic connections typically increases as $O(N^2)$. Since the activation

function is extremely sensitive to small changes in temperature and that there are far fewer neuronal parameters to update than synaptic weights, suggests that the adaptive neuron model should offer a significant reduction in convergence time.

In this paper we have also shown that the active participation of the neurons during the supervised learning phase led to a significant reduction in the number of training cycles required to learn logic type of problems. In the adaptive neuron model both the synaptic weight interconnection strengths and the neuronal temperature terms are treated as optimization parameters and have their own updating scheme and time scales. This learning rule is based on implementing gradient descent in the sum squared error $E$ with respect to both the weights $w_{ij}^m$ and temperatures $T_i^m$. Preliminary results indicate that the new algorithm can significantly outperform back propagation by reducing the learning time by several orders of magnitude. Specifically, the XOR problem was learnt to a very high precision by the network in $\approx 10^3$ training iterations with a mean square error of $\approx 10^{-6}$ versus over 106 iterations with a corresponding mean square error of $\approx 10^{-3}$.

### Acknowledgements.

The work described in this paper was performed by the Jet Propulsion Laboratory, California Institute of Technology, and was supported in parts by the National Aeronautics and Space Administration and the Defense Advanced Research Projects Agency through an agreement with the National Aeronautics and Space Administration.

REFERENCES

1. D. Rummelhart, J. McClelland, "Parallel Distributed Processing," M.I.T. Press, Cambridge, MA, 1986.
2. J. J. Hopfield, *Neural Networks as Physical Systems with Emergent Collective Computational Abilities*, Proceedings of the National Academy of Science USA **79** (1982), 2554–2558.
3. F. J. Pineda, *Generalization Of Backpropagation To Recurrent and Higher Order Neural Networks*, in "Neural Information Processing Systems Proceedings," AIP, New York, 1987.
4. L. R. Carley, *Presynaptic Neural Information Processing*, in "Neural Information Processing Systems Proceedings," AIP, New York, 1987.
